# A Topographic Support Vector Machine: Classification Using Local Label Configurations

**Johannes Mohr**
Clinic for Psychiatry and Psychotherapy
Charité Medical School
and
Bernstein Center for Computational Neuroscience Berlin
10117 Berlin, Germany

**Klaus Obermayer**
Department of Electrical Engineering and Computer Science
Berlin University of Technology
and
Bernstein Center for Computational Neuroscience Berlin
10587 Berlin, Germany

johann@cs.tu-berlin.de, oby@cs.tu-berlin.de

## Abstract

The standard approach to the classification of objects is to consider the examples as independent and identically distributed (iid). In many real world settings, however, this assumption is not valid, because a topographical relationship exists between the objects. In this contribution we consider the special case of image segmentation, where the objects are pixels and where the underlying topography is a 2D regular rectangular grid. We introduce a classification method which not only uses measured vectorial feature information but also the label configuration within a topographic neighborhood. Due to the resulting dependence between the labels of neighboring pixels, a collective classification of a set of pixels becomes necessary. We propose a new method called 'Topographic Support Vector Machine' (TSVM), which is based on a topographic kernel and a self-consistent solution to the label assignment shown to be equivalent to a recurrent neural network. The performance of the algorithm is compared to a conventional SVM on a cell image segmentation task.

## 1   Introduction

The segmentation of natural images into semantically meaningful subdivisions can be considered as one or more binary pixel classification problems, where two classes of pixels are characterized by some measurement data (features). For each binary problem the task is to assign a set of new pixels to one of the two classes using a classifier trained on a set of labeled pixels (training data).

In conventional classification approaches usually the assumption of iid examples is made, so the classification result is determined solely by the measurement data. Natural images, however, possess a topographic structure, in which there are dependencies between the labels of topographic neighbors, making the data non-iid. Therefore, not only the measurement data, but also the labels of the topographic neighbors can be used in the classification of a pixel. It has been shown for a number of problems that dependencies between instances can improve model accuracy. A Conditional Random Field approach approach has been used for labeling text sequences by [1]. Combining this idea with local discriminative models, in [2] a discriminative random field was used to model the dependencies between the labels of image blocks in a probabilistic framework. A collective classification relational dependency network was used in [3] for movie box-office receipts prediction and paper topic classification. The maximization of the per label margin of pairwise Markov networks was applied in [4] to handwritten character recognition and collective hypertext classification. There, the number of variables and constraints of the quadratic programming problem was polynomial in the number of labels.

In this work, we propose a method which is also based on margin maximization and allows the collective assignments of a large number of binary labels which have a regular grid topography. In contrast to [4] the number of constraints and variables does not depend on the number of labels. The method called topographic support vector machine (TSVM) is based on the assumption that knowledge about the local label configuration can improve the classification of a single data point. Consider as example the segmentation of a collection of images depicting physical objects of similar shape, but high variability in gray level and texture. In this case, the measurements are dissimilar, while the local label configurations show high similarity.

Here, we apply the TSVM to the supervised bottom-up segmentation of microscopic images of Papanicolaou stained cervical cell nuclei from the CSSIP pap smear dataset[1]. Segmentation of these images is important for the detection of cervical cancer or precancerous cells. The final goal is to use so-called malignancy associated changes (MACs), e.g. a slight shift of the distribution of nuclear size not yet visual to the human observer, in order to detect cancer at an early stage [5]. A previously used bottom-up segmentation approach for this data using morphological watersheds was reported to have difficulties with weak gradients and the presence of other large gradients adjacent to the target [5]. Top-down methods like active contour models have successfully been used [6], but require heuristic initialization and error correction procedures.

## 2   Classification using a Topographic Support Vector Machine

Let $\mathcal{O} = \{o_1, ..., o_n\}$ be a set of $n$ sites on a 2D pixel-grid and $\mathcal{G} = \{\mathcal{G}_o, o \in \mathcal{O}\}$ be a neighborhood system for $\mathcal{O}$, where $\mathcal{G}_o$ is the set of neighbors of $o$ and neighborhood is defined by $o \notin \mathcal{G}_o$ and $o \in \mathcal{G}_p \Leftrightarrow p \in \mathcal{G}_o$. For each pixel site $o_i$ from the set $\mathcal{O}$, a binary label $y_i \in \{-1, +1\}$ giving the class assignment is assumed to be known. To simplify the notation, in the following we are going to make use of multi-indices written in the form of vectors, referring to pairs of indices on a two-dimensional grid. We define the neighborhood of order c as $\mathcal{G}^c = \{\mathcal{G}_\mathbf{i}, \mathbf{i} \in \mathcal{O}\}; \mathcal{G}_\mathbf{i} = \{\mathbf{k} \in \mathcal{O} : 0 < (\mathbf{k} - \mathbf{i})^2 \le c\}$. This way, $\mathcal{G}^1$ describes the first order neighborhood system (4 neighbors), $\mathcal{G}^2$ the second order system (8 neighbors), and so on. Each pixel site is characterized by some measurement vector. This could for example be the vector of gray value intensities at a pixel site, the gray value patch around a central pixel location, or the responses to a bank of linear or nonlinear filters (e.g. Gabor coefficients). Using a training set composed of (possibly several) sets of pixel sites, each accompanied by a set of measurement vectors $X = \{\mathbf{x}_i, \forall i \in [1..n]\}$ and a set of

labels $Y = \{y_i, \forall i \in [1..n]\}$ (e.g. a manually labeled image), the task of classification is to assign class labels to a set of $\kappa$ pixels sites $\mathcal{U} = \{u_1, ..., u_\kappa\}$ of an unlabeled image, for which a set of measurements $\tilde{X} = \{\tilde{\mathbf{x}}_i, \forall i \in [1..\kappa]\}$ is available. For the classification we will use a support vector machine.

## 2.1 Support Vector Classification

In Support Vector Classification (SVC) methods ([7]), a kernel is used to solve a complex classification task in a usually high-dimensional feature space via a separating hyperplane. Results from statistical learning theory ([8]) show that maximizing the margin (the distance of the closest data point to the hyperplane) leads to improved generalization abilities. In practice, the optimal margin hyperplane can be obtained solving a quadratic programming problem. Several schemes have been introduced to deal with noisy measurements via the introduction of slack variables. In the following we will shortly review one such scheme, the C-SVM, which is also later used in the experiments. For a canonical separating hyperplane $(\mathbf{w}, b)$ in a higher dimensional feature space $\mathcal{H}$, to which the $n$ variables $\mathbf{x_i}$ are mapped by $\phi(\mathbf{x})$, and $n$ slack variables $\xi_i$ the primal objective function of a C-SVM can be formulated as

$$\min_{\mathbf{w} \in \mathcal{H}, \boldsymbol{\xi} \in \mathbb{R}^n} \left( \frac{1}{2} \|\mathbf{w}\|^2 + \frac{C}{n} \sum_{i=1}^{n} \xi_i \right), \tag{1}$$

subject to $y_i(\mathbf{w}^T \phi(\mathbf{x_i}) + b) \leq 1 - \xi_i, \quad \xi_i \geq 0, \quad C > 0, \quad i = 1, ..., n.$

In order to classify a new object $h$ with unknown label, the following decision rule is evaluated:

$$f(\mathbf{x_h}) = \text{sgn}\Big( \sum_{i=1}^{m} \alpha_i y_i K(\mathbf{x_h}, \mathbf{x_i}) + b \Big), \tag{2}$$

where the sum runs over all m support vectors.

## 2.2 Topographic Kernel

We now assume that the label of each pixel is determined by both the measurement and the set of labels of its topographic neighbors. We define a vector $\mathbf{y}_{\mathcal{G}_h}$ where the labels of the $q$ topographic neighbors of the pixel $\mathbf{h}$ are concatenated in an arbitrary, but fixed order. We propose a support vector classifier using an extended kernel, which in addition to the measurement vector $\mathbf{x_h}$, also includes the vector $\mathbf{y}_{\mathcal{G}_h}$:

$$K(\mathbf{x_h}, \mathbf{x_j}, \mathbf{y}_{\mathcal{G}_h}, \mathbf{y}_{\mathcal{G}_j}) = K_1(\mathbf{x_h}, \mathbf{x_j}) + \lambda \cdot K_2(\mathbf{y}_{\mathcal{G}_h}, \mathbf{y}_{\mathcal{G}_j}), \tag{3}$$

where $\lambda$ is a hyper-parameter. Kernel $K_1$ can be an arbitrary kernel working on the measurements. For kernel $K_2$ an arbitrary dot-product kernel might be used. In the following we restrict ourselves to a linear kernel (corresponding to the normalized Hamming distance between the local label configurations)

$$K_2(\mathbf{y}_{\mathcal{G}_h}, \mathbf{y}_{\mathcal{G}_j}) = \frac{1}{q} \langle \mathbf{y}_{\mathcal{G}_h} | \mathbf{y}_{\mathcal{G}_j} \rangle, \tag{4}$$

where $\langle ...|... \rangle$ denotes a scalar product. The kernel $K_2$ defined in eq. (4) thus consists of a dot-product between these vectors divided by their length. For a neighborhood $\mathcal{G}_\mathbf{h}^c$ of order $c$ we obtain

$$K_2(\mathbf{y}_{\mathcal{G}_h}, \mathbf{y}_{\mathcal{G}_j}) = \frac{1}{q} \sum_{|\mathbf{s}| < \sqrt{c}, \mathbf{s} \neq \mathbf{0}} y_{\mathbf{h}+\mathbf{s}} \cdot y_{\mathbf{j}+\mathbf{s}} \tag{5}$$

The linear kernel $K_2$ in (4) takes on its maximum value, if the label configurations are identical, and its lowest value if the label configuration is inverted.

### 2.3 Learning phase

If a SVM is trained using the topographic kernel (3), the topographic label configuration is included in the learning process. The resulting support vectors will still contain the relevant information about the measurements, but additionally the label neighborhood information relevant for a good distinction of the classes.

### 2.4 Classification phase

In order to collectively classify a set of $\kappa$ new pixel sites with unknown topographic label configuration, we propose the following iterative approach to achieve a self-consistent solution to the classification problem. We denote the labels at step $\tau$ as $y_{\mathbf{h}}(\tau), \forall \mathbf{h}$. At each step $\tau$ new labels are assigned according to

$$y_{\mathbf{h}}(\tau) = \operatorname{sgn}\Big( \sum_{j=1}^{m} \alpha_j \cdot y_{\mathbf{v}(j)} \cdot K(\mathbf{x_h}, \mathbf{x}_{\mathbf{v}(j)}, \mathbf{y}_{\mathcal{G}_{\mathbf{h}}}(\tau-1), \mathbf{y}_{\mathcal{G}_{\mathbf{v}(j)}}) + b \Big), \forall \mathbf{h}. \qquad (6)$$

The sum runs over all $m$ support vectors, whose indices on the 2D grid are denoted by the vector $\mathbf{v}(j)$. Since initially the labels are unknown, we use at step $\tau = 0$ the results from a conventional support-vector machine ($\lambda = 0$) as initialization for the labels. For the following steps some estimates of the neighboring labels are available from the previous iteration. Using this new topographic label information in addition to the measurement information, using (6) a new assignment decision for the labels is made. This leads to an iterative assignment of new labels.

If we write the contributions from kernel $K_1$, which depend only on $x$ and do not change with $\tau$, as $c_{\mathbf{h}}(j) = \alpha_j \cdot y_{\mathbf{v}(j)} \cdot K_1(\mathbf{x_h}, \mathbf{x}_{\mathbf{v}(j)})$ equation (6) becomes

$$y_{\mathbf{h}}(\tau) = \operatorname{sgn}\Big( \sum_{j=1}^{m} \big[ \lambda \alpha_j y_{\mathbf{v}(j)} K_2(\mathbf{y}_{\mathcal{G}_{\mathbf{h}}}(\tau-1), \mathbf{y}_{\mathcal{G}_{\mathbf{v}(j)}}) + c_{\mathbf{h}}(j) \big] + b \Big), \forall \mathbf{h}. \qquad (7)$$

Putting in the linear kernel from equation (5), we get

$$y_{\mathbf{h}}(\tau) = \operatorname{sgn}\Big( \sum_{j=1}^{m} \big[ \alpha_j y_{\mathbf{v}(j)} \frac{\lambda}{q} \sum_{|\mathbf{s}|<\sqrt{c}, \mathbf{s} \neq \mathbf{0}} [y_{\mathbf{h+s}}(\tau-1) \cdot y_{\mathbf{v}(j)+\mathbf{s}}] + c_{\mathbf{h}}(j) \big] + b \Big), \forall \mathbf{h}. \qquad (8)$$

Interchanging the sums, using the definitions

$$w_{\mathbf{h,k}} = \begin{cases} \frac{\lambda}{q} \sum_{j=1}^{m} \alpha_j y_{\mathbf{v}(j)} y_{\mathbf{v}(j)+(\mathbf{k-h})} & : \quad \mathbf{k} \in \mathcal{G}_{\mathbf{h}} \\ 0 & : \quad \mathbf{k} \notin \mathcal{G}_{\mathbf{h}} \end{cases} \qquad (9)$$

and

$$\theta_{\mathbf{h}} = -\Big( \sum_{j=1}^{m} c_{\mathbf{h}}(j) + b \Big), \qquad (10)$$

we obtain

$$y_{\mathbf{h}}(\tau) = \operatorname{sgn}\Big( \sum_{\mathbf{k}} y_{\mathbf{k}}(\tau-1) \cdot w_{\mathbf{h,k}} - \theta_{\mathbf{h}} \Big), \forall \mathbf{h}. \qquad (11)$$

This corresponds to the equations describing the dynamics of a recurrent neural network composed of McCulloch-Pitts neurons [9]. The condition for symmetric weights $w_{\mathbf{h,k}} = w_{\mathbf{k,h}}$ is equivalent to an inversion symmetry of the label configurations of the support vectors in the neighborhood topology, therefore the weights in equation (9) are not necessarily symmetric. A suitable stopping criterion for the iteration is that the net reaches either a fixed point $y_{\mathbf{h}}(\tau) = y_{\mathbf{h}}(\tau-1), \forall \mathbf{h}$, or an attractor cycle $y_{\mathbf{h}}(\tau) = y_{\mathbf{h}}(\rho), \rho < \tau-1, \forall \mathbf{h}$.

The network described by eq. (11) corresponds to a diluted network of $\kappa$ binary neurons with no self-interaction and asymmetric weights. One can see from eq.(9) that the network has only local connections, corresponding to the topographic neighborhood $\mathcal{G}_\mathbf{h}$. The measurement $\mathbf{x_h}$ only influences the individual unit threshold $\theta_\mathbf{h}$ of the network, via the weighted sum over all support vectors of the contributions from kernel $K_1$ (eq. (10)). The label configurations of the support vectors, on the other hand, are contained in the network weights via eq.(9). The weights are multiplied by the hyper-parameter $\lambda$, which determines how much the label configuration influences the class decision in comparison to the measurements. It has to be adjusted to yield optimal results for a class of data sets. For $\lambda = 0$ the TSVM becomes a conventional SVM.

## 2.5 Symmetrization of the weights

In order to ensure convergence, we suggest to use an inversion symmetric version $K_2^{sym}$ of kernel $K_2$. For the pixel grid we can define the inversion operation as $\mathbf{l} + \mathbf{t} \rightarrow \mathbf{l} - \mathbf{t}, \quad \mathbf{t} \in \mathbb{N}^2, \quad \forall \mathbf{l} + \mathbf{t} \in \mathcal{G}_\mathbf{l}$, and denote the inverse of $\mathbf{a}$ by $\bar{\mathbf{a}}$. Taking the inverse of the vector $\mathbf{y}_{\mathcal{G}_\mathbf{l}}$, in which the set $y_{\mathcal{G}_\mathbf{l}}$ is concatenated in an arbitrary but fixed order, leads to a reordering of the components of the vector. The benefit from the chosen inversion symmetric kernel is that the self consistency equations for the labels will turn out to be equivalent to a Hopfield net, which has proven convergence properties. We define the new kernel as

$$K_2^{sym}(\mathbf{y}_{\mathcal{G}_\mathbf{h}}, \mathbf{y}_{\mathcal{G}_\mathbf{j}}) = \frac{1}{q}\left(\langle \mathbf{y}_{\mathcal{G}_\mathbf{h}}|\mathbf{y}_{\mathcal{G}_\mathbf{j}}\rangle + \langle \mathbf{y}_{\mathcal{G}_\mathbf{h}}|\bar{\mathbf{y}}_{\mathcal{G}_\mathbf{j}}\rangle\right). \tag{12}$$

Although only the second argument is inverted within the kernel, the value of this kernel does not depend on the order of the arguments.

**Proof**  It follows from the definition of the inversion operator and the dot product that $\langle \mathbf{y}_{\mathcal{G}_\mathbf{h}}|\mathbf{y}_{\mathcal{G}_\mathbf{j}}\rangle = \langle \bar{\mathbf{y}}_{\mathcal{G}_\mathbf{h}}|\bar{\mathbf{y}}_{\mathcal{G}_\mathbf{j}}\rangle = \langle \mathbf{y}_{\mathcal{G}_\mathbf{j}}|\mathbf{y}_{\mathcal{G}_\mathbf{h}}\rangle = \langle \bar{\mathbf{y}}_{\mathcal{G}_\mathbf{j}}|\bar{\mathbf{y}}_{\mathcal{G}_\mathbf{h}}\rangle$ and $\langle \bar{\mathbf{y}}_{\mathcal{G}_\mathbf{h}}|\mathbf{y}_{\mathcal{G}_\mathbf{j}}\rangle = \langle \mathbf{y}_{\mathcal{G}_\mathbf{h}}|\bar{\mathbf{y}}_{\mathcal{G}_\mathbf{j}}\rangle = \langle \mathbf{y}_{\mathcal{G}_\mathbf{j}}|\bar{\mathbf{y}}_{\mathcal{G}_\mathbf{h}}\rangle = \langle \bar{\mathbf{y}}_{\mathcal{G}_\mathbf{j}}|\mathbf{y}_{\mathcal{G}_\mathbf{h}}\rangle$. Therefore,

$$
\begin{aligned}
K_2^{sym}(\mathbf{y}_{\mathcal{G}_\mathbf{h}}, \mathbf{y}_{\mathcal{G}_\mathbf{j}}) &= \frac{1}{q}\left(\langle \mathbf{y}_{\mathcal{G}_\mathbf{h}}|\mathbf{y}_{\mathcal{G}_\mathbf{j}}\rangle + \langle \mathbf{y}_{\mathcal{G}_\mathbf{h}}|\bar{\mathbf{y}}_{\mathcal{G}_\mathbf{j}}\rangle\right) = \frac{1}{q}\left(\langle \mathbf{y}_{\mathcal{G}_\mathbf{j}}|\mathbf{y}_{\mathcal{G}_\mathbf{h}}\rangle + \langle \bar{\mathbf{y}}_{\mathcal{G}_\mathbf{j}}|\mathbf{y}_{\mathcal{G}_\mathbf{h}}\rangle\right) \\
&= \frac{1}{q}\left(\langle \mathbf{y}_{\mathcal{G}_\mathbf{j}}|\mathbf{y}_{\mathcal{G}_\mathbf{h}}\rangle + \langle \mathbf{y}_{\mathcal{G}_\mathbf{j}}|\bar{\mathbf{y}}_{\mathcal{G}_\mathbf{h}}\rangle\right) = K_2^{sym}(\mathbf{y}_{\mathcal{G}_\mathbf{j}}, \mathbf{y}_{\mathcal{G}_\mathbf{h}}) \qquad \square.
\end{aligned}
$$

Putting kernel (12) into eq.(7) and defining

$$w_{\mathbf{h},\mathbf{k}}^{sym} = \begin{cases} \frac{\lambda}{q}\sum_{j=1}^{m}\alpha_j y_{\mathbf{v}(j)}\left(y_{\mathbf{v}(j)+(\mathbf{k}-\mathbf{h})} + y_{\mathbf{v}(j)-(\mathbf{k}-\mathbf{h})}\right) & : \quad \mathbf{k} \in \mathcal{G}_\mathbf{h} \\ 0 & : \quad \mathbf{k} \notin \mathcal{G}_\mathbf{h} \end{cases} \tag{13}$$

we get

$$y_\mathbf{h}(\tau) = \text{sgn}\left(\sum_\mathbf{k} y_\mathbf{k}(\tau-1)\cdot w_{\mathbf{h},\mathbf{k}}^{sym} - \theta_\mathbf{h}\right), \forall \mathbf{h}. \tag{14}$$

Since the network weights $w_{\mathbf{h},\mathbf{j}}^{sym}$ defined in eq.(13) are symmetric this corresponds to the equation describing the dynamics during the retrieval phase of a Hopfield network [10]. Instead of taking the sum over all patterns, the sum is taken over all support vectors. The weight between two neurons in the original Hopfield net corresponds to the correlation between two components (over all fundamental patterns). In (13) the weight only depends on the difference vector $\mathbf{k}$-$\mathbf{h}$ between the two neurons on the 2D grid and is proportional to the correlation (over all support vectors) between the label of a support vector and the label in the distance $\mathbf{k}$-$\mathbf{h}$.

Table 1: Average misclassification rate $R$ and the standard deviation of the mean $\sigma$ at optimal hyper-parameters $C$, $S$ and $\lambda$.

| algorithm | $\log_2 C$ | $\log_2 S$ | $\lambda$ | $R[\%]$ | $\sigma[\%]$ |
|-----------|------------|------------|-----------|---------|--------------|
| SVM | 4 | 0.5 | 0 | 2.23 | 0.05 |
| STSVM | 4 | 0.5 | 1.2 | 1.96 | 0.06 |
| TSVM | 2 | 0.5 | 1.4 | 1.86 | 0.05 |

## 3   Experiments

We applied the above algorithms to the binary classification of pixel sites of cell images from the CSSIP pap smear dataset. The goal was to assign the label +1 to the pixels belonging to the nucleus, and -1 to all others. The dataset contains three manual segmentations of the nucleus' boundaries, from which we generated a 'ground truth' label for the area of the nucleus using a majority voting. Only the first 300 images were used in the experiments. As a measurement vector we took a column-ordering of a 3x3 gray value patch centered on a pixel site. In order to measure the classification performance for a non-iid data set, we estimated the test error based on the collective classification of all pixels in several randomly chosen test images. We compared three algorithms: A conventional SVM, the 'TSVM' with the topographic kernel $K_2$ from eq.(4) and the 'STSVM' with the inversion symmetric topographic kernel $K_2^{sym}$ from eq.(12). In the experiments we used a label neighborhood of order 32, which corresponds to $q = 100$ neighbors. For kernel $K_1$ we used an RBF kernel $K_1(\mathbf{x_1}, \mathbf{x_2}) = \exp(-\|\mathbf{x_1} - \mathbf{x_2}\|^2/S^2)$ with hyper-parameter $S$. Since the data set was very large, no cross-validation or resampling techniques were required, and only a small subset of the available training data could be used for training. We randomly sampled several disjoint training sets in order to improve the accuracy of the error estimation. First, the hyper-parameters $S$ and $C$ (for TSVM and STSVM also $\lambda$) were optimized via a grid search in parameter space. This was done by measuring the average test error over 20 test images and 5 training sets. Then, the test of the classifiers was conducted at the in each case optimal hyper-parameters for 20 yet unused test images and 50 randomly sampled disjoint training sets. In all experiments using synchronous update either a fixed point or an attractor cycle of length two was reached. The average number of iterations needed was 12 (TSVM) and 13 (STSVM). Although the convergence properties have only been formally proven for the symmetric weight STSVM, experimental evidence suggests the same convergence properties for the TSVM. The results for synchronous update are shown in table 1 (results using asynchronous update differed only by 0.01%). The performance of both topographic algorithms is superior to the conventional SVM, while the TSVM performed slightly better than the STSVM. For the top-down method in [6] the results were only qualitatively assessed by a human expert, not quantitatively compared to a manual segmentation, therefore a direct comparison to our results was not possible. To illustrate the role of the hyper-parameter $\lambda$, fig.1 shows 10 typical test images and their segmentations achieved by an STSVM at different values of $\lambda$ for fixed $S$ and $C$. For increasing $\lambda$ the label images become less noisy, and at $\lambda = 0.4$ most artifacts have disappeared. This is caused by the increasing weight put on the label configuration via kernel $K_2^{sym}$. Increasing $\lambda$ even further will eventually lead to the appearance of spurious artifacts, as the influence of the label configuration will dominate the classification decision.

## 4   Conclusions

We have presented a classification method for a special case of non-iid data in which the objects are linked by a regular grid structure. The proposed algorithm is composed of two

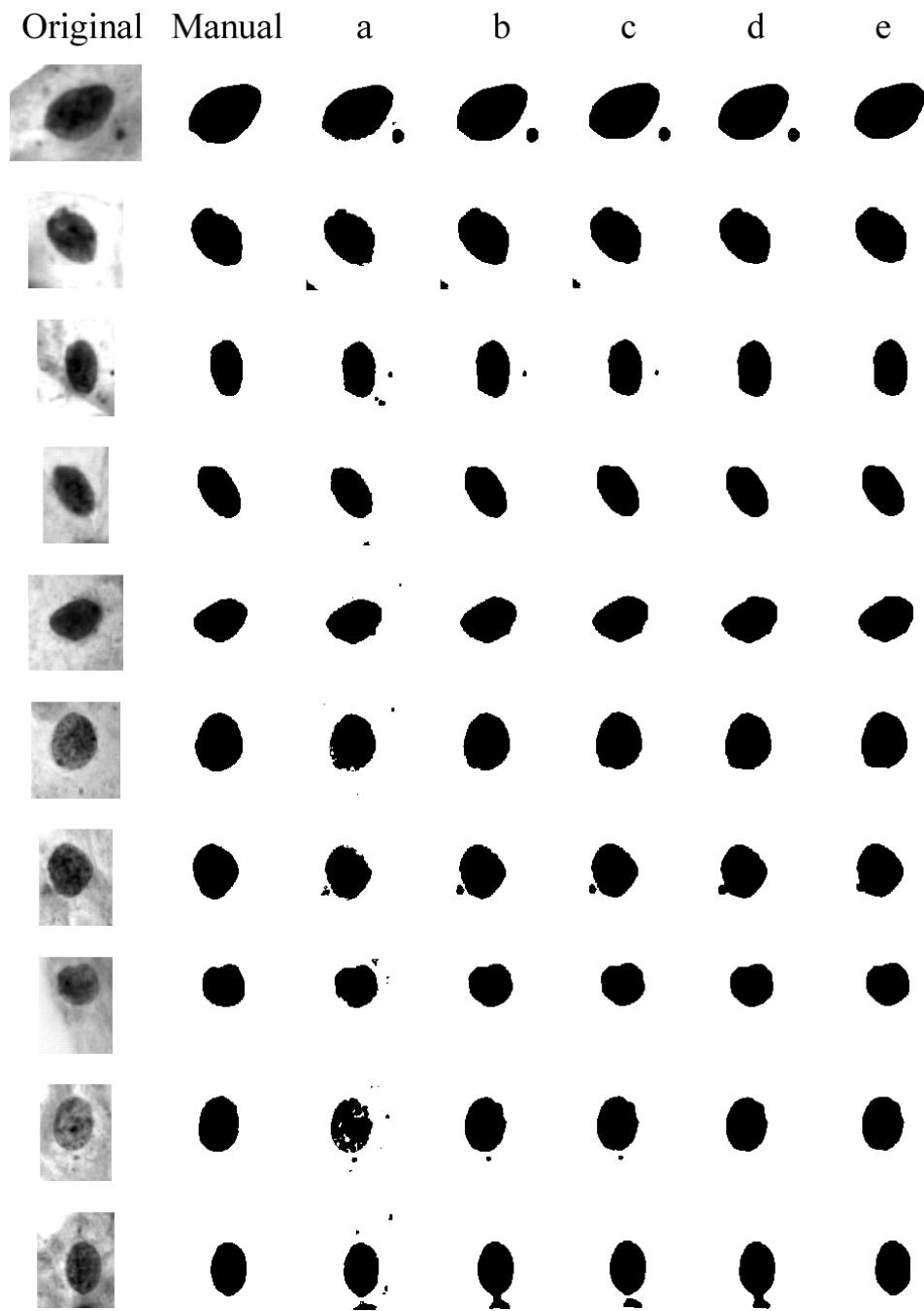

Figure 1: Final labels assigned by the STSVM at fixed hyper-parameters $C = 2^6, S = 2^2$. (a) $\lambda = 0$, (b) $\lambda = 0.1$, (c) $\lambda = 0.2$, (d) $\lambda = 0.3$, (e) $\lambda = 0.4$.

components: The first part is a topographic kernel which integrates conventional feature information and the information of the label configurations within a topographic neighborhood. The second part consists of a collective classification with recurrent neural network dynamics which lets local label configurations converge to attractors determined by the label configurations of the support vectors. For the asymmetric weight TSVM, the dimensionality of the problem is increased by the neighborhood size as compared to a conventional SVM (twice the neighborhood size for the symmetric weight STSVM). However, the number of variables and constraints does not increase with the number of data points to be labeled. Therefore, the TSVM and the STSVM can be applied to image segmentation problems, where a large number of pixel labels have to be assigned simultaneously.

The algorithms were applied to the bottom-up cell nucleus segmentation in pap smear images needed for the detection of cervical cancer. The classification performance of the TSVM and STSVM were compared to a conventional SVM, and it was shown that the inclusion of the topographic label configuration lead to a substantial decrease in the average misclassification rate. The two topographic algorithms were much more resistant to noise and smaller artifacts. A removal of artifacts which have similar size and the same measurement features as some of the nuclei cannot be achieved by a pure bottom-up method, as this requires a priori model knowledge. In practice, the lower dimensional TSVM is to be preferred over the STSVM, since it is faster and performed slightly better.

### Acknowledgments

This work was funded by the BMBF (grant 01GQ0411). We thank Sepp Hochreiter for useful discussions.

## Footnotes

[1]Centre for Sensor Signal and Information Processing, University of Queensland

# References

[1] J. Lafferty; A. McCallum; F. Pereira. Conditional random fields: Probabilistic models for segmenting and labeling sequence data. In *Proc. Int. Conf. on Machine Learning*, 2001.

[2] S. Kumar; M. Hebert. Discriminative fields for modeling spatial dependencies in natural images. In Sebastian Thrun, Lawrence Saul, and Bernhard Schölkopf, editors, *Advances in Neural Information Processing Systems 16*. MIT Press, Cambridge, MA, 2004.

[3] J. Neville and D. Jensen. Collective classification with relational dependency networks. In *Proc. 2nd Multi-Relational Data Mining Workshop, 9th ACM SIGKDD Intern. Conf. Knowledge Discovery and Data Mining*, 2003.

[4] B. Taskar, C. Guestrin, and D. Koller. Max-margin markov networks. In Sebastian Thrun, Lawrence Saul, and Bernhard Schölkopf, editors, *Advances in Neural Information Processing Systems 16*. MIT Press, Cambridge, MA, 2004.

[5] P. Bamford. *Segmentation of Cell Images with Application to Cervical Cancer Screening*. PhD thesis, University of Queensland, 1999.

[6] P. Bamford and B. Lovell. Unsupervised cell nucleus segmentation with active contours. *Signal Processing Special Issue: Deformable Models and Techniques for Image and Signal Processing*, 71(2):203–213, 1998.

[7] B. Schölkopf and A. Smola. *Learning with Kernels*. The MIT Press, 2002.

[8] V. Vapnik. *Statistical Learning Theory*. Springer, New York, 1998.

[9] W. McCulloch and W. Pitts. A logical calculus of the ideas immanent in nervous activity. *Bulletin of mathematical physics*, 5:115–133, 1943.

[10] S. Haykin. *Neural Networks*. Macmillan College Publishing Company Inc., 1994.